# Learning with Tree-Averaged Densities and Distributions

**Sergey Kirshner**
AICML and Dept of Computing Science
University of Alberta
Edmonton, Alberta, Canada T6G 2E8
`sergey@cs.ualberta.ca`

## Abstract

We utilize the ensemble of trees framework, a tractable mixture over super-exponential number of tree-structured distributions [1], to develop a new model for multivariate density estimation. The model is based on a construction of tree-structured copulas – multivariate distributions with uniform on $[0, 1]$ marginals. By averaging over all possible tree structures, the new model can approximate distributions with complex variable dependencies. We propose an EM algorithm to estimate the parameters for these tree-averaged models for both the real-valued and the categorical case. Based on the tree-averaged framework, we propose a new model for joint precipitation amounts data on networks of rain stations.

## 1   Introduction

Multivariate real-valued data appears in many real-world data sets, and a lot of research is being focused on the development of multivariate real-valued distributions. One of the challenges in constructing such distributions is that univariate continuous distributions commonly do not have a clear multivariate generalization. The most studied exception is the multivariate Gaussian distribution owing to properties such as closed form density expression with a convenient generalization to higher dimensions and closure over the set of linear projections. However, not all problems can be addressed fairly with Gaussians (e.g., mixtures, multimodal distributions, heavy-tailed distributions), and new approaches are needed for such problems.

While modeling multivariate distributions is in general difficult due to complicated functional forms and the curse of dimensionality, learning models for individual variables (univariate marginals) is often straightforward. Once the univariate marginals are known (or assumed known), the rest can be modeled using *copulas*, multivariate distributions with all univariate marginals equal to uniform distributions on $[0, 1]$ (e.g., [2, 3]). A large portion of copula research concentrated on bivariate copulas as extensions to higher dimensions are often difficult. Thus if the desired distribution decomposes into its univariate marginals and only bivariate distributions, the machinery of copulas can be effectively utilized.

Distributions with undirected tree-structured graphical models (e.g., [4]) have exactly these properties, as probability density functions over the variables with tree-structured conditional independence graphs can be written as a product involving univariate marginals and bivariate marginals corresponding to the edges of the tree. While tree-structured dependence is perhaps too restrictive, a richer variable dependence can be obtained by averaging over a small number of different tree structures [5] or *all* possible tree structures; the latter can be done analytically for categorical-valued distributions with an ensemble-of-trees model [1]. In this paper, we extend this tree-averaged model to continuous variables with the help of copulas and derive a learning algorithm to estimate the parameters within the maximum likelihood framework with EM [6]. Within this framework, the

parameter estimation for tree-structured and tree-averaged models requires optimization over only univariate and bivariate densities potentially avoiding the curse of dimensionality, a property not shared by alternative models that relax the dependence restriction of trees (e.g., vines [7]).

The main contributions of the paper are the new tree-averaged model for multivariate copulas, a parameter estimation algorithm for tree-averaged framework (for both categorical and real-valued complete data), and a new model for multi-site daily precipitation amounts, an important application in hydrology. In the process, we introduce previously unexplored tree-structured copula density and an algorithm for estimation of its structure and parameters. The paper is organized as follows. First, we describe copulas, their densities, and some of their useful properties (Section 2). We then construct multivariate copulas with tree-structured dependence from bivariate copulas (Section 3.1) and show how to estimate the parameters of the bivariate copulas and perform the edge selection. To allow more complex dependencies between the variables, we describe a *tree-averaged copula*, a novel copula object constructed by averaging over all possible spanning trees for tree-structured copulas, and derive a learning algorithm for the estimation of the parameters from data for the tree-averaged copulas (Section 4). We apply our new method to a benchmark data set (Section 5.1); we also develop a new model for multi-site precipitation amounts, a problem involving both binary (rain/no rain) and continuous (how much rain) variables (Section 5.2).

## 2 Copulas

Let $\boldsymbol{X} = (X_1, \ldots, X_d)$ be a vector random variable with corresponding probability distribution $F$ (cdf) defined on $\mathbb{R}^d$. We denote by $\mathcal{V}$ the set of $d$ components (variables) of $\boldsymbol{X}$ and refer to individual variables as $X_v$ for $v \in \mathcal{V}$. For simplicity, we will refer to assignments to random variables by lower case letters, e.g., $X_v = x_v$ will be denoted by $x_v$. Let $F_v(x_v) = F(X_v = x_v, X_u = \infty : u \in \mathcal{V} \setminus \{v\})$ denote a univariate marginal of $F$ over the variable $X_v$. Let $p_v(x_v)$ denote the probability density function (pdf) of $X_v$. Let $a_v = F_v(x_v)$, and let $\boldsymbol{a} = (a_1, \ldots, a_d)$, so $\boldsymbol{a}$ is a vector of quantiles of components of $\boldsymbol{x}$ with respect to corresponding univariate marginals. Next, we define copula, a multivariate distribution over vectors of quantiles.

**Definition 1.** The **copula** associated with $F$ is a distribution function $C : [0,1]^d \to [0,1]$ that satisfies

$$F(\boldsymbol{x}) = C(F_1(x_1), \ldots, F_d(x_d)), \ \boldsymbol{x} \in \mathbb{R}^d. \tag{1}$$

If $F$ is a continuous distribution on $\mathbb{R}^d$ with univariate marginals $F_1, \ldots, F_d$, then $C(\boldsymbol{a}) = F\left(F_1^{-1}(a_1), \ldots, F_d^{-1}(a_d)\right)$ is the unique choice for (1).

Assuming that $F$ has $d$-th order partial derivatives, the probability density function (pdf) can be obtained from the distribution function via differentiation and expressed in terms of a derivative of a copula:

$$p(\boldsymbol{x}) = \frac{\partial^d F(\boldsymbol{x})}{\partial x_1 \ldots \partial x_d} = \frac{\partial^d C(\boldsymbol{a})}{\partial x_1 \ldots \partial x_d} = \frac{\partial^d C(\boldsymbol{a})}{\partial a_1 \ldots \partial a_d} \prod_{v \in \mathcal{V}} \frac{\partial a_v}{\partial x_v} = c(\boldsymbol{a}) \prod_{v \in \mathcal{V}} p_v(x_v) \tag{2}$$

where $c(\boldsymbol{a}) = \frac{\partial^d C(\boldsymbol{a})}{\partial a_1 \ldots \partial a_d}$ is referred to as a *copula density function*.

Suppose we are given a complete data set $\mathcal{D} = \left\{\boldsymbol{x}^1, \ldots, \boldsymbol{x}^N\right\}$ of $d$-component real-valued vectors $\boldsymbol{x}^n = \left(x_1^n, \ldots, x_1^d\right)$ under i.i.d. assumption. A maximum likelihood (ML) estimate for the parameters of $c$ (or $p$) from data can be obtained my maximizing the log-likelihood of $\mathcal{D}$

$$\ln p(\mathcal{D}) = \sum_{v \in \mathcal{V}} \sum_{n=1}^N \ln p_v(x_v^n) + \sum_{n=1}^N \ln c(F_1(x_1^n), \ldots, F_d(x_d^n)). \tag{3}$$

The first term of the log-likelihood corresponds to the total log-likelihood of all univariate marginals of $p$, and the second term to the log-likelihood of its $d$-variate copula. These terms are not independent as the second term in the sum is defined in terms of the probability expressions in the first summand; except for a few special cases, a direct optimization of (3) is prohibitively complicated. However a useful (and asymptotically consistent) heuristic is first to maximize the log-likelihood for the marginals (first term only), and then to estimate the parameters for the copula given the solution

for the marginals. The univariate marginals can be accurately estimated by either fitting the parameters for some appropriately chosen univariate distributions or by applying non-parametric methods[1] as the marginals are estimated independent of each other and do not suffer from the curse of dimensionality. Let $\hat{p}_v(x_v)$ be the estimated pdf for component $v$, and $\hat{F}_v$ be the corresponding cdf. Let $\mathcal{A} = \{\boldsymbol{a}^1, \ldots, \boldsymbol{a}^N\}$ where $\boldsymbol{a}^n = (a_1^n, \ldots, a_d^n) = \left(\hat{F}(x_1^n), \ldots, \hat{F}(x_d^n)\right)$ be a set of estimated quantiles. Under the above heuristic, ML estimate for copula density $c$ is computed by maximizing $\ln c(\mathcal{A}) = \sum_{n=1}^N \ln c(\boldsymbol{a}^n)$.

## 3  Exploiting Tree-Structured Dependence

Joint probability distributions are often modeled with probabilistic graphical models where the structure of the graph captures the conditional independence relations of the variables. The joint distribution is then represented as a product of functions over subsets of variables. We would like to keep the number of variables for each of the functions small as the number of parameters and the number of points needed for parameter estimation often grows exponentially with the number of variables. Thus, we focus on copulas with tree dependence. Trees play an important role in probabilistic graphical models as they allow for efficient exact inference [10] as well as structure and parameter learning [4]. They can also be placed in a fully Bayesian framework with decomposable priors allowing to compute expected values (over all possible spanning trees) of product of functions defined on the edges of the trees [1]. As we will see later in this section, under the tree-structured dependence, a copula density can be computed as products of bivariate copula densities over the edges of the graph. This property allows us to estimate the parameters for the edge copulas independently.

### 3.1  Tree-Structured Copulas

We consider tree-structured Markov networks, i.e., undirected graphs that do not have loops. For a distribution $F$ admitting tree-structured Markov networks (referred from now on as tree-structured distributions), assuming that $p(\boldsymbol{x}) > 0$ and $p(\boldsymbol{x}) < \infty$ for $\boldsymbol{x} \in \mathcal{R} \subseteq \mathbb{R}^d$, the density (for $\boldsymbol{x} \in \mathcal{R}$) can be rewritten as

$$p(\boldsymbol{x}) = \left[\prod_{v \in \mathcal{V}} p_v(x_v)\right] \prod_{\{u,v\} \in \mathcal{E}} \frac{p_{uv}(x_u, x_v)}{p_u(x_u) p_v(x_v)}. \tag{4}$$

This formulation easily follows from the Hammersley-Clifford theorem [11]. Note that for $\{u, v\} \in \mathcal{E}$, a copula density $c_{uv}(a_u, a_v)$ for $F(x_u, x_v)$ can be computed using Equation 2:

$$c_{uv}(a_u, a_v) = \frac{p_{uv}(x_u, x_v)}{p_u(x_u) p_v(x_v)}. \tag{5}$$

Using Equations 2, 4, and 5, $c_p(\boldsymbol{a})$ for $F(\boldsymbol{x})$ can be computed as

$$c_p(\boldsymbol{a}) = \frac{p(\boldsymbol{x})}{\prod_{v \in \mathcal{V}} p_v(x_v)} = \prod_{\{u,v\} \in \mathcal{E}} \frac{p_{uv}(x_u, x_v)}{p_u(x_u) p_v(x_v)} = \prod_{\{u,v\} \in \mathcal{E}} c_p(a_u, a_v). \tag{6}$$

Equation 6 states that a copula density for a tree-structured distribution decomposes as a product of bivariate copulas over its edges. The converse is true as well; a tree-structured copula can be constructed by specifying copulas for the edges of the tree.

**Theorem 1.** Given a tree or a forest $\mathcal{G} = (\mathcal{V}, \mathcal{E})$ and copula densities $c_{uv}(a_u, a_v)$ for $\{u, v\} \in \mathcal{E}$,

$$c_{\mathcal{E}}(\boldsymbol{a}) = \prod_{\{u,v\} \in \mathcal{E}} c_{uv}(a_u, a_v)$$

is a valid copula density.

For a tree-structured density, the copula log-likelihood can be rewritten as

$$\ln c(\mathcal{A}) = \sum_{\{u,v\} \in \mathcal{E}} \sum_{n=1}^N \ln c_{uv}(a_u^n, a_v^n),$$

and the parameters can be fitted by maximizing $\sum_{n=1}^{N} \ln c_{uv}\left(a_u^n, a_v^n\right)$ independently for different pairs $\{u, v\} \in \mathcal{E}$. The tree structure can be learned from the data as well, as in the Chow-Liu algorithm [4]. Full algorithm can be found in an extended version of the paper [12].

## 4  Tree-Averaged Copulas

While the framework from Section 3.1 is computationally efficient and convenient for implementation, the imposed tree-structured dependence is too restrictive for real-world problems. Vines [7], for example, deal with this problem by allowing recursive refinements for the bivariate probabilities over variables not connected by the tree edges. However, vines require estimation of additional characteristics of the distribution (e.g., conditional rank correlations) requiring estimation over large sets of variables, which is not advisable when the amount of available data is not large. Our proposed method would only require optimization of parameters of bivariate copulas from the corresponding two components of weighted data vectors. Using the Bayesian framework for spanning trees from [1], it is possible to construct an object constituting a convex combination over *all* possible spanning trees allowing a much richer set of conditional independencies than a single tree.

Meilă and Jaakkola [1] proposed a decomposable prior over all possible spanning tree structures. Let $\boldsymbol{\beta}$ be a symmetric matrix of non-negative weights for all pairs of distinct variables and zeros on the diagonal. Let $\mathfrak{E}$ be a set of all possible spanning trees over $\mathcal{V}$. The probability distribution over all spanning tree structures over $\mathcal{V}$ is defined as

$$P\left(\mathcal{E} \in \mathfrak{E} | \boldsymbol{\beta}\right) = \frac{1}{Z} \prod_{\{u,v\} \in \mathcal{E}} \beta_{uv} \text{ where } Z = \sum_{\mathcal{E} \in \mathfrak{E}} \prod_{\{u,v\} \in \mathcal{E}} \beta_{uv}. \tag{7}$$

Even though the sum is over $|\mathfrak{E}| = d^{d-2}$ trees, $Z$ can be efficiently computed in closed form using a weighted generalization of Kirchoff's Matrix Tree Theorem (e.g., [1]).

**Theorem 2.** Let $P\left(\mathcal{E}\right)$ be a distribution over spanning tree structures defined by (7). Then the normalization constant $Z$ is equal to the determinant $\left|\mathbf{L}^{\star}\left(\boldsymbol{\beta}\right)\right|$, with matrix $\mathbf{L}^{\star}\left(\boldsymbol{\beta}\right)$ representing the first $(d-1)$ rows and columns of the matrix $\mathbf{L}\left(\boldsymbol{\beta}\right)$ given by:

$$L_{uv}\left(\boldsymbol{\beta}\right) = L_{vu}\left(\boldsymbol{\beta}\right) = \begin{cases} -\beta_{uv} & u, v \in \mathcal{V}, \ u \neq v; \\ \sum_{w \in \mathcal{V}} \beta_{vw} & u, v \in \mathcal{V}, \ u = v. \end{cases}$$

$\boldsymbol{\beta}$ is a generalization of an adjacency matrix, and $\mathbf{L}\left(\boldsymbol{\beta}\right)$ is a generalization of the Laplacian matrix. The decomposability property of the tree prior (Equation 7) allows us to compute the average of the tree-structured distributions over all $d^{d-2}$ tree structures. In [1], such averaging was applied to tree-structured distributions over categorical variables. Similarly, we define a *tree-averaged copula* density as a convex combination of copula densities of the form (6):

$$r\left(\boldsymbol{a}\right) = \sum_{\mathcal{E} \in \mathfrak{E}} P\left(\mathcal{E} | \boldsymbol{\beta}\right) c\left(\boldsymbol{a}\right) = \frac{1}{Z} \sum_{\mathcal{E} \in \mathfrak{E}} \left[ \prod_{\{u,v\} \in \mathcal{E}} \beta_{uv} \right] \left[ \prod_{\{u,v\} \in \mathcal{E}} c_{uv}\left(a_u, a_v\right) \right] = \frac{\left|\mathbf{L}^{\star}\left(\boldsymbol{\beta} \boldsymbol{c}\left(a\right)\right)\right|}{\left|\mathbf{L}^{\star}\left(\boldsymbol{\beta}\right)\right|}$$

where entry $(uv)$ of matrix $\boldsymbol{\beta} \boldsymbol{c}\left(\boldsymbol{a}\right)$ denotes $\beta_{uv} c_{uv}\left(a_u, a_v\right)$. A finite convex combination of copulas is a copula, so $r\left(\boldsymbol{a}\right)$ is a copula density.

### 4.1  Parameter Estimation

Given a set of estimated quantile values $\mathcal{A}$, a suitable parameter values $\boldsymbol{\beta}$ (edge weight matrix) and $\boldsymbol{\theta}$ (parameters for bivariate edge copulas) can be found by maximizing the log-likelihood of $\mathcal{A}$:

$$l\left(\boldsymbol{\beta}, \boldsymbol{\theta}\right) = \ln r\left(\mathcal{A} | \boldsymbol{\beta}, \boldsymbol{\theta}\right) = \sum_{n=1}^{N} \ln r\left(\boldsymbol{a}^n | \boldsymbol{\beta}, \boldsymbol{\theta}\right) = \sum_{n=1}^{N} \ln \left|L^{\star}\left(\boldsymbol{\beta} \boldsymbol{c}\left(\boldsymbol{a}^n | \boldsymbol{\theta}\right)\right)\right| - N \ln \left|L^{\star}\left(\boldsymbol{\beta}\right)\right|. \tag{8}$$

However, the parameter optimization of $l\left(\boldsymbol{\beta}, \boldsymbol{\theta}\right)$ cannot be done analytically. Instead, noticing that we are dealing with a mixture model (granted, one where the number of mixture components is super-exponential), we propose performing the parameter optimization with the EM algorithm [6].[2]

Algorithm TREEAVERAGEDCOPULADENSITY$(\mathcal{D}, \boldsymbol{c})$

**Inputs:** A complete data set $\mathcal{D}$ of $d$-component real-valued vectors; a set of of bivariate parametric copula densities $\boldsymbol{c} = \{c_{uv} : u, v \in \mathcal{V}\}$

1. Estimate univariate margins $\hat{F}_v(X_v)$ for all components $v \in \mathcal{V}$ treating all components independently.

2. Replace $\mathcal{D}$ with $\mathcal{A}$ consisting of vectors $\boldsymbol{a}^n = \left(\hat{F}_1(x_1^n), \ldots, \hat{F}_d(x_d^n)\right)$ for each vector $\boldsymbol{x}^n$ in $\mathcal{D}$

3. Initialize $\boldsymbol{\beta}$ and $\boldsymbol{\theta}$

4. Run until convergence (as determined by change in log-likelihood, Equation 8)
   - E-step: For all vectors $a^n$ and pairs $\{u, v\}$, compute $P(\{u, v\} \in \mathcal{E} | \boldsymbol{a}^n, \boldsymbol{\beta}, \boldsymbol{\theta})$
   - M-step:
     - Update $\boldsymbol{\beta}$ with gradient ascent
     - Update $\boldsymbol{\theta}_{uv}$ for all pairs by setting partial derivative with respect to parameters of $\boldsymbol{\theta}_{uv}$ (Equation 9) to zero and solving corresponding equations

**Output:** Denoting $a_u = \hat{F}(x_u)$ and $a_v = \hat{F}(x_v)$, $\hat{p}(\boldsymbol{x}) = \left[\prod_{v \in \mathcal{V}} \hat{p}_v(x_v)\right] \frac{|\mathbf{L}^\star(\boldsymbol{\beta}\boldsymbol{c}(\boldsymbol{a}))|}{|\mathbf{L}^\star(\boldsymbol{\beta})|}$

---

Figure 1: Algorithm for estimation of a pdf with tree-averaged copulas.

While there are $d^{d-2}$ possible mixture components (spanning trees), in the E-step, we only need to compute the posterior probabilities for $d(d-1)/2$ edges. Each step of EM consists of finding parameters $\boldsymbol{\beta}', \boldsymbol{\theta}'$ maximizing the expected joint log-likelihood $M(\boldsymbol{\beta}', \boldsymbol{\theta}'; \boldsymbol{\beta}, \boldsymbol{\theta})$ given current parameter values $\boldsymbol{\beta}, \boldsymbol{\theta}$ where

$$
\begin{aligned}
M(\boldsymbol{\beta}', \boldsymbol{\theta}'; \boldsymbol{\beta}, \boldsymbol{\theta}) &= \sum_{n=1}^N \sum_{\mathcal{E}_n \in \mathfrak{E}} P(\mathcal{E}_n | \boldsymbol{a}^n, \boldsymbol{\beta}, \boldsymbol{\theta}) \ln\left[P(\mathcal{E}|\boldsymbol{\beta}') c(\boldsymbol{a}^n | \mathcal{E}, \boldsymbol{\theta}')\right] \\
&= \sum_{\{u,v\}} \sum_{n=1}^N s_n(\{u, v\}) \left(\ln \beta'_{uv} + \ln c_{uv}(a_u^n, a_v^n | \theta'_{uv})\right) - N \ln |\mathbf{L}^\star(\boldsymbol{\beta}')|; \\
s_n(\{u, v\}) &= \sum_{\substack{\mathcal{E} \in \mathfrak{E} \\ \{u,v\} \in \mathcal{E}}} P(\mathcal{E}_n | \boldsymbol{a}^n, \boldsymbol{\beta}, \boldsymbol{\theta}) = \sum_{\substack{\mathcal{E} \in \mathfrak{E} \\ \{u,v\} \in \mathcal{E}}} \frac{\prod_{\{u,v\} \in \mathcal{E}} (\beta_{uv} c_{uv}(a_u^n, a_v^n | \boldsymbol{\theta}_{uv}))}{|L^\star(\boldsymbol{\beta}\boldsymbol{c}(\boldsymbol{a}^n))|}.
\end{aligned}
$$

The probability distribution $P(\mathcal{E}_n | \boldsymbol{a}^n, \boldsymbol{\beta}, \boldsymbol{\theta})$ is of the same form as the tree prior, so to compute $s_n(\{u, v\})$ one needs to compute the sum of probabilities of all trees containing edge $\{u, v\}$.

**Theorem 3.** Let $P(\mathcal{E}|\boldsymbol{\beta})$ be a tree prior defined in Equation 7. Let $\mathbf{Q}(\boldsymbol{\beta}) = (\mathbf{L}^\star(\boldsymbol{\beta}))^{-1}$ where $\mathbf{L}^\star$ is obtained by removing row and column $w$ from $\mathbf{L}$. Then

$$
\sum_{\mathcal{E} \in \mathfrak{E}: \{u,v\} \in \mathcal{E}} P(\mathcal{E}|\boldsymbol{\beta}) = \begin{cases} \beta_{uv}(Q_{uu}(\boldsymbol{\beta}) + Q_{vv}(\boldsymbol{\beta}) - 2Q_{uv}(\boldsymbol{\beta})) & : \quad u \neq v, u \neq w, v \neq w, \\ \beta_{uw} Q_{uu}(\boldsymbol{\beta}) & : \quad v = w, \\ \beta_{wv} Q_{vv}(\boldsymbol{\beta}) & : \quad u = w. \end{cases}
$$

As a consequence of Theorem 3, for each $\boldsymbol{a}^n$, all $d(d-1)/2$ edge probabilities $s_n(\{u, v\})$ can be computed *simultaneously* with time complexity of a single $(d-1) \times (d-1)$ matrix inversion, $\mathcal{O}(d^3)$. Assuming a candidate bivariate copula $c_{uv}$ has one free parameter $\theta_{uv}$, $\theta_{uv}$ can be optimized by setting

$$
\frac{\partial M(\boldsymbol{\beta}', \boldsymbol{\theta}'; \boldsymbol{\beta}, \boldsymbol{\theta})}{\partial \theta'_{uv}} = \sum_{n=1}^N s_n(\{u, v\}) \frac{\partial \ln c_{uv}(a_u^n, a_v^n; \theta'_{uv})}{\partial \theta'_{uv}}, \tag{9}
$$

to 0. (See [12] for more details.) The parameters of the tree prior can be updated by maximizing

$$
\sum_{\{u,v\}} \left(\frac{1}{N} \sum_{n=1}^N s_n(\{u, v\})\right) \ln \beta'_{uv} - \ln |\mathbf{L}^\star(\boldsymbol{\beta})|,
$$

an expression concave in $\ln \beta_{uv} \ \forall \{u, v\}$. $\boldsymbol{\beta}'$ can be updated using a gradient ascent algorithm on $\ln \beta_{uv} \ \forall \{u, v\}$, with time complexity $\mathcal{O}\left(d^3\right)$ per iteration. The outline of the EM algorithm is shown in Figure 1. Assuming the complexity of each bivariate copula update is $\mathcal{O}\left(N\right)$, the time complexity of each EM iteration is $\mathcal{O}\left(Nd^3\right)$.

The EM algorithm can be easily transferred to tree averaging for categorical data. The E-step does not change, and in the M-step, the parameters for the univariate marginals are updated ignoring bivariate terms. Then, the parameters for the bivariate distributions for each edge are updated constrained on the new values of the parameters for the univariate distributions. While the algorithm does not guarantee a maximization of the expected log-likelihood, it nonetheless worked well in our experiments.

# 5 Experiments

## 5.1 MAGIC Gamma Telescope Data Set

First, we tested our tree-averaged density estimator on a MAGIC Gamma Telescope Data Set from the UCI Machine Learning Repository [13]. We considered only the examples from class gamma (signal); this set consists of 12332 vectors of $d = 10$ real-valued components. The univariate marginals are not Gaussian (some are bounded; some have multiple modes). Fig. 2 shows an average log-likelihood of models trained on training sets with $N = 50, 100, 200, 500, 1000, 2000, 5000, 10000$ and evaluated on 2000-example test sets (averaged over 10 training and test sets). The marginals were estimated using Gaussian kernel density estimators (KDE) with Rule-of-Thumb bandwidth selection. All of the models except for full Gaussian have the same marginals, differ only in the multivariate dependence (copula). As expected from the curse of dimensionality, product KDE improves logarithmically with the amount of data. Not only the marginals are not Gaussian (evidenced by a Gaussian copula with KDE marginals outperforming a Gaussian distribution), the multivariate dependence is also not Gaussian, evidenced by a tree-structured Frank copula outperforming a tree-structured and a full Gaussian copula. However, model averaging even with the wrong dependence model (tree-averaged Gaussian copula) yields superior performance.

## 5.2 Multi-Site Precipitation Modeling

We applied the tree-averaged framework to the problem of modeling daily rainfall amounts for a regional spatial network of stations. The task is to build a generative model capturing the spatial and temporal properties of the data. This model can be used in at least two ways: first, to sample sequences from it and to use them as inputs for other models, e.g., crop models; and second, as a descriptive model of the data. Hidden Markov models (possible with non-homogeneous transitions) are being frequently used for this task (e.g., [14]) with the transition distribution responsible for modeling of temporal dependence, and the emission distributions capturing most of the spatial dependence. Additionally, HMMs can be viewed as assigning rainfall daily patterns to "weather states" (or corresponding emission components), and both these states (as described by either their parameters or the statistics of the patterns associated with it) and their temporal evolution often offer useful synoptic insight. We will use HMMs as the wrapper model with tree-averaged (and tree-structured) distributions to model the emission components.

The distribution of daily rainfall amounts for any given station can be viewed as a non-overlapping mixture with one component corresponding to zero precipitation, and the other component to positive precipitation. For a station $v$, let $r_v$ be the precipitation amount, $\pi_v$ be a probability of positive precipitation, and let $f_v\left(r_v|\boldsymbol{\lambda}_v\right)$ be a probability density function for amounts given positive precipitation:

$$p\left(r_v|\pi_v, \boldsymbol{\lambda}_v\right) = \left\{ \begin{array}{ll} 1 - \pi_v & : \quad r_v = 0, \\ \pi_v f_v\left(r_v|\boldsymbol{\lambda}_v\right) & : \quad r_v > 0. \end{array} \right.$$

For a pair of stations $\{u, v\}$, let $\pi_{uv}$ denote the probability of simultaneous positive amounts and $c_{uv}\left(F_u\left(r_u|\boldsymbol{\lambda}_u\right), F_v\left(r_v|\boldsymbol{\lambda}_v\right)|\boldsymbol{\theta}_{uv}\right)$ denote the copula density for simultaneous positive amounts;

then

$$
p\left(r_u, r_v | \pi_u, \pi_v, \pi_{uv}, \boldsymbol{\lambda}_u, \boldsymbol{\lambda}_v\right) = \begin{cases}
1 - \pi_u - \pi_v + \pi_{uv} & : \quad r_u = 0, \; r_v = 0, \\
\left(\pi_v - \pi_{uv}\right) f_v\left(r_v | \boldsymbol{\lambda}_v\right) & : \quad r_u = 0, \; r_v > 0, \\
\left(\pi_u - \pi_{uv}\right) f_u\left(r_u | \boldsymbol{\lambda}_u\right) & : \quad r_u > 0, \; r_v = 0, \\
\pi_{uv} f_u\left(r_u\right) f_v\left(r_v\right) c\left(F_u\left(r_u\right), F_v\left(r_v\right)\right) & : \quad r_u > 0, \; r_v > 0.
\end{cases}
$$

We can now define a tree-structured and tree-averaged probability distributions, $p_t\left(\boldsymbol{r}\right)$ and $p_{ta}\left(\boldsymbol{r}\right)$, respectively, over the amounts:

$$
\omega_{uv}\left(\boldsymbol{r}\right) = \frac{p\left(r_u, r_v | \pi_u, \pi_v, \pi_{uv}, \boldsymbol{\lambda}_u, \boldsymbol{\lambda}_v\right)}{p\left(r_u | \pi_u, \boldsymbol{\lambda}_u\right) p\left(r_v | \pi_v, \boldsymbol{\lambda}_v\right)}, \quad p_t\left(\boldsymbol{r} | \boldsymbol{\pi}, \boldsymbol{\lambda}, \boldsymbol{\theta}, \mathcal{E}\right) = \left[\prod_{v \in \mathcal{V}} p\left(r_v | \pi_v\right)\right] \prod_{\{u,v\} \in \mathcal{E}} \omega_{uv}\left(\boldsymbol{r}\right),
$$

$$
p_{ta}\left(\boldsymbol{r} | \boldsymbol{\pi}, \boldsymbol{\lambda}, \boldsymbol{\theta}, \boldsymbol{\beta}\right) = \sum_{\mathcal{E} \in \mathfrak{E}} P\left(\mathcal{E} | \boldsymbol{\beta}\right) p_t\left(\boldsymbol{r} | \boldsymbol{\pi}, \boldsymbol{\lambda}, \boldsymbol{\theta}, \mathcal{E}\right) = \left[\prod_{v \in \mathcal{V}} p\left(r_v | \pi_v\right)\right] \frac{\left|\mathbf{L}^\star\left(\boldsymbol{\beta}\boldsymbol{\omega}\left(\boldsymbol{r}\right)\right)\right|}{\left|\mathbf{L}^\star\left(\boldsymbol{\beta}\right)\right|}.
$$

We employ univariate exponential distributions $f_v\left(r_v\right) = \lambda_v e^{-\lambda_v r_v}$ and bivariate Gaussian copulas

$$
c_{uv}\left(a_u, a_v\right) = \frac{1}{\sqrt{1 - \theta_{uv}^2}} e^{-\frac{\theta_{uv}^2 \Phi^{-1}(a_u)^2 + \theta_{uv}^2 \Phi^{-1}(a_v)^2 - 2\theta_{uv}\Phi^{-1}(a_u)\Phi^{-1}(a_v)}{2\left(1 - \theta_{uv}^2\right)}}.
$$

We applied the models to a data set collected from 30 stations from a region in Southeastern Australia (Fig. 3) 1986-2005, April-October, (20 sequences 214 30-dimensional vectors each). We used a 5-state HMM with three different types of emission distributions: tree-averaged ($p_{ta}$), tree-structured ($p_t$), and conditionally independent (first term of $p_t$ and $p_{ta}$). We will refer to these models HMM-TA, HMM-Tree, and HMM-CI, respectively. For HMM-TA, we reduced the number of free parameters by only allowing edges for stations adjacent to each other as determined by the the Delaunay triangulation (Fig. 3). We also did not learn the edge weights ($\boldsymbol{\beta}$) setting them to 1 for selected edges and to 0 for the rest. To make sure that the models do not overfit, we computed their out-of-sample log-likelihood with cross-validation, leaving out one year at a time (not shown). (5 states were chosen because the leave-one-year out log-likelihood starts to flatten out for HMM-TA at 5 states.) The resulting log-likelihoods divided by the number of days and stations are $-0.9392$, $-0.9522$, and $-1.0222$ for HMM-TA, HMM-Tree, and HMM-CI, respectively. To see how well the models capture the properties of the data, we trained each model on the whole data set (with 50 restarts of EM), and then simulated 500 sequences of length 214. We are particularly interested in how well they measure pairwise dependence; we concentrate on two measures: log-odds ratio for occurrence and Kendall's $\tau$ measure of concordance for pairs when both stations had positive amounts. Both are shown in Fig. 4. Both plots suggest that HMM-CI underestimates the pairwise dependence for strongly dependent pairs (as indicated by its trend to predict lower absolute values for log-odds and concordance); HMM-Tree estimating the dependence correctly mostly for strongly dependent pairs (as indicated by good prediction for high values), but underestimating it for moderate dependence; and HMM-TA performing the best for most pairs except for the ones with very strong dependence.

## Acknowledgements

This work has been supported by the Alberta Ingenuity Fund through the AICML. We thank Stephen Charles (CSIRO, Australia) for providing us with precipitation data.

## Footnotes

[1] These approaches for copula estimation are referred to as *inference for the margins* (IFM) [8] and *canonical maximum likelihood* (CML) [9] for parametric and non-parametric forms for the marginals, respectively.

[2] A possibility of EM algorithm for ensemble-of-trees with categorical data was mentioned [1], but the idea was abandoned due to the concern about the M-step.

## References

[1] M. Meilă and T. Jaakkola. Tractable Bayesian learning of tree belief networks. *Statistics and Computing*, 16(1):77–92, 2006.

[2] H. Joe. *Multivariate Models and Dependence Concepts*, volume 73 of *Monographs on Statistics and Applied Probability*. Chapman & Hall/CRC, 1997.

[3] R. B. Nelsen. *An Introduction to Copulas*. Springer Series in Statistics. Springer, 2nd edition, 2006.

[4] C. K. Chow and C. N. Liu. Approximating discrete probability distributions with dependence trees. *IEEE Transactions on Information Theory*, IT-14(3):462–467, May 1968.

[5] M. Meilă and M. I. Jordan. Learning with mixtures of trees. *Journal of Machine Learning Research*, 1(1):1–48, October 2000.

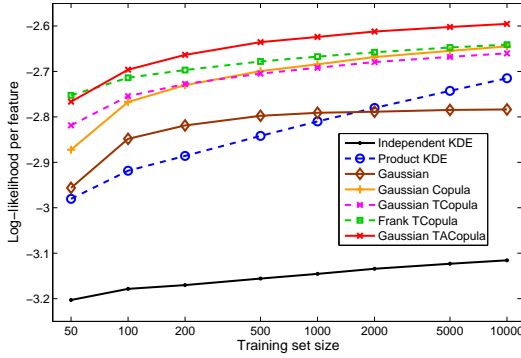

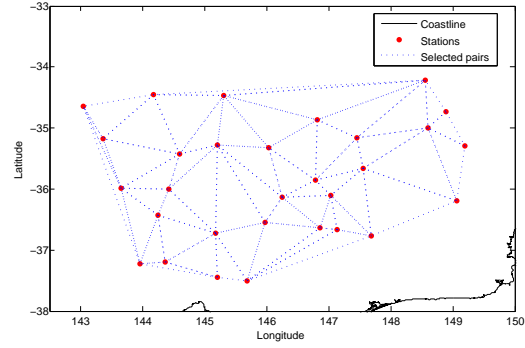

Figure 2: Averaged test set per-feature log-likelihood for MAGIC data: independent KDE (black solid .), product KDE (blue dashed ○), Gaussian (brown solid ◇), Gaussian copula (orange solid +), Gaussian tree-copula (magenta dashed x), Frank tree-copula (blue dashed □), Gaussian tree-averaged copula (red solid x).

Figure 3: Station map with station locations (red dots), coastline, and the pairs of stations selected according to Delaunay triangulation (dotted lines)

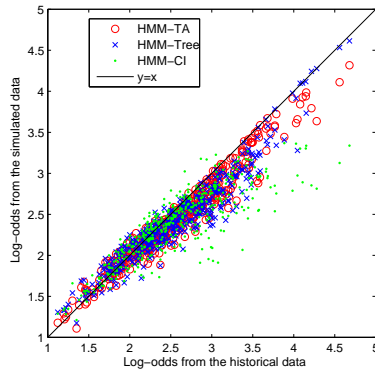

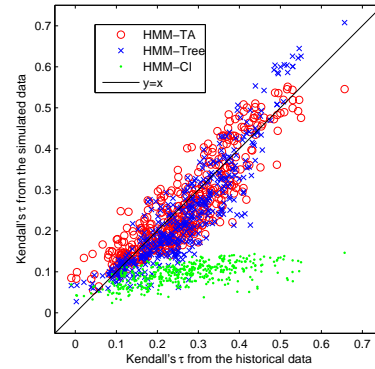

Figure 4: Scatter-plots of log-odds ratios for occurrence (left) and Kendall's $\tau$ measure of concordance (right) for all pairs of stations for the historical data vs HMM-TA (red o), HMM-Tree (blue x), and HMM-CI (green ·).

[6] A. P. Dempster, N. M. Laird, and D. B. Rubin. Maximum likelihood from incomplete data via EM algorithm. *Journal of the Royal Statistical Society Series B-Methodological*, 39(1):1–38, 1977.

[7] T. Bedford and R. M. Cooke. Vines – a new graphical model for dependent random variables. *The Annals of Statistics*, 30(4):1031–1068, 2002.

[8] H. Joe and J.J. Xu. The estimation method of inference functions for margins for multivariate models. Technical report, Department of Statistics, University of British Columbia, 1996.

[9] C. Genest, K. Ghoudi, and L.-P. Rivest. A semiparametric estimation procedure of dependence parameters in multivariate families of distributions. *Biometrika*, 82:543–552, 1995.

[10] J. Pearl. *Probabilistic Reasoning in Intelligent Systems: Networks of Plausible Inference*. Morgan Kaufmann Publishers, Inc., San Francisco, California, 1988.

[11] J. Besag. Spatial interaction and the statistical analysis of lattice systems. *Journal of the Royal Statistical Society Series B-Methodological*, 36(2):192–236, 1974.

[12] S. Kirshner. Learning with tree-averaged densities and distributions. Technical Report TR 08-01, Department of Computing Science, University of Alberta, 2008.

[13] A. Asuncion and D.J. Newman. UCI machine learning repository, 2007.

[14] E. Bellone. *Nonhomogeneous Hidden Markov Models for Downscaling Synoptic Atmospheric Patterns to Precipitation Amounts*. PhD thesis, Department of Statistics, University of Washington, 2000.

